# Biologically Plausible Local Learning Rules for the Adaptation of the Vestibulo-Ocular Reflex

**Olivier Coenen***
Computational Neurobiology Laboratory
Howard Hughes Medical Institute
The Salk Institute
P.O.Box 85800
San Diego, CA 92186-5800

**Terrence J. Sejnowski**

**Stephen G. Lisberger**
Department of Physiology
W.M. Keck Foundation Center
for Integrative Neuroscience
University of California,
San Fransisco, CA, 94143

## Abstract

The vestibulo-ocular reflex (VOR) is a compensatory eye movement that stabilizes images on the retina during head turns. Its magnitude, or gain, can be modified by visual experience during head movements. Possible learning mechanisms for this adaptation have been explored in a model of the oculomotor system based on anatomical and physiological constraints. The local correlational learning rules in our model reproduce the adaptation and behavior of the VOR under certain parameter conditions. From these conditions, predictions for the time course of adaptation at the learning sites are made.

## 1   INTRODUCTION

The primate oculomotor system is capable of maintaining the image of an object on the fovea even when the head and object are moving simultaneously. The vestibular organs provide information about the head velocity with a short delay of 14 ms but visual signals from the moving object are relatively slow and can take 100 ms to affect eye movements. The gain, $G$, of the VOR, defined as minus the eye velocity over the head velocity $(-\dot{e}/\dot{h})$, can be modified by wearing magnifying or diminishing glasses (figure 1). VOR adaptation, absent in the dark, is driven by the combination of image slip on the retina and head turns.

*University of California, San Diego. Dept. of Physics. La Jolla, CA, 92037. Email address: olivier@helmholtz.sdsc.edu

During head turns on the first day of wearing magnifying glasses, the magnified image of an object slips on the retina. After a few days of adaptation, the eye velocity and hence the gain of the VOR increases to compensate for the image magnification.

We have constructed a model of the VOR and smooth pursuit systems that uses biologically plausible local learning rules that are consistent with anatomical pathways and physiological recordings. The learning rules in the model are local in the sense that the adaptation of a synapse depends solely on signals that are locally available. A similar model with different local learning rules has been recently proposed (Quinn *et al.*, Neuroscience 1992).

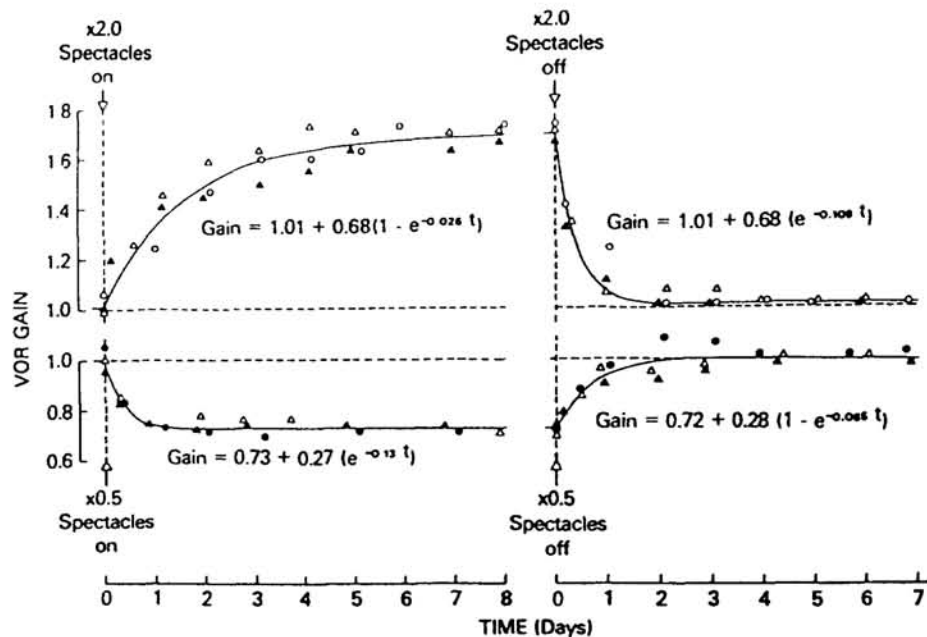

Figure 1: Time course of the adapting VOR and its recovery of gain in monkeys exposed to the long-term influence of magnifying (upper curves) and diminishing (lower curves) spectacles. Different symbols obtained from different animals, demonstrating the consistency of the adaptive change. From Melvill Jones (1991), selected from Miles and Eighmy (1980).

## 2   THE MODEL

Feedforward and recurrent models of the VOR have been proposed (Fujita, 1982; Galiana, 1986; Kawato and Gomi, 1992; Quinn et al., 1992; Arnold and Robinson, 1992; Lisberger and Sejnowski, 1992). In this paper we study a static and linear version of a previously studied recurrent network model of the VOR and smooth pursuit system (Lisberger, 1992; Lisberger and Sejnowski, 1992; Viola, Lisberger and Sejnowski, 1992). The time delays and time constants associated with nodes in the network were eliminated so that the time course of the VOR plasticity could be more easily analyzed (figure 2).

The model describes the system ipsilateral to one eye. The visual error, which carries the image retinal slip velocity signal, is a measure of the performance of both the VOR and smooth pursuit system as well as the main error signal for learning. The value at each node represents changes in its firing rate from its resting firing rate. The transformation from the rate of firing of premotor signal ($N$) to eye velocity is represented in the model by a gain

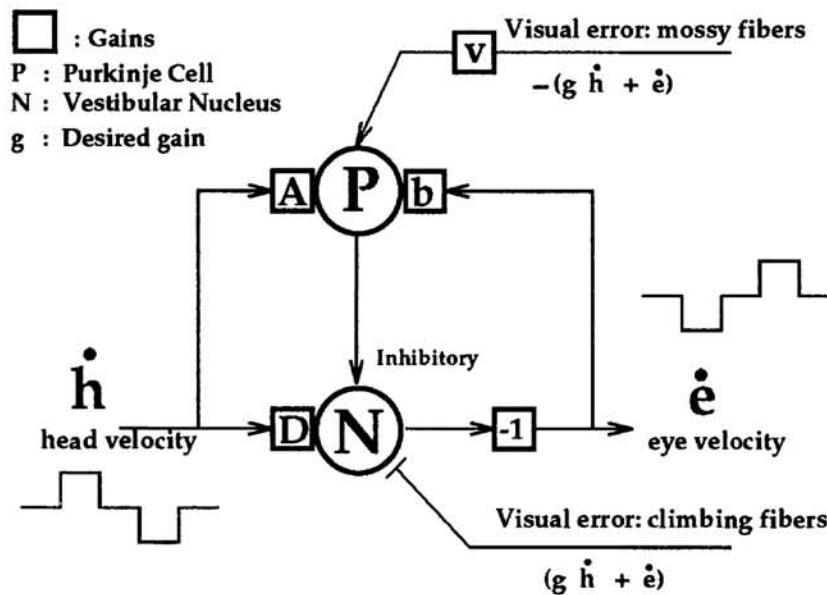

Figure 2: **Diagram of the VOR and smooth pursuit model.** The input and output of the model are, respectively, head velocity and eye velocity. The model has three main parts: the node $P$ represents an ensemble of Purkinje cells from the ventral paraflocculus of the cerebellum, the node $N$ represents an ensemble of flocculus-target neurons in the vestibular nucleus, and the visual inputs which provide the visual error signals in the mossy and climbing fibers. The capital letter gains $A$ and $D$, multiplying the input signals to the nodes, are modified according to their learning rules. The lower case letters $b$, $v$, and $g$ are also multiplicative gains, but remain constant during adaptation. The traces represent head and eye velocity modulation in time. The visual error signal in the climbing fibers drives learning in node $N$ but does not constitute one of its inputs in the present model.

of $-1$. The gain of the VOR in this model is given by $\frac{D-A}{1-b}$. We have not modeled the neural integrator that converts eye velocity commands to eye position signals that drive the motoneurons.

## 3  LEARNING RULES

We have adopted the learning rules proposed by Marr (1969), Albus (1971) and Ito (1970) for adaptation in the cerebellum and by Lisberger (1988), Miles and Lisberger (1981) for plasticity in the brain stem (figure 3). These are variations of the delta rule and depend on an explicit representation of the error signal at the synapses.

Long term depression at mossy fiber synapses on Purkinje cells has been observed *in vitro* under simultaneous stimulation of climbing fibers and mossy fibers (Ito, Sakurai and Tongroach, 1982). In addition, we have included a learning mechanism for potentiation of mossy fiber head velocity inputs under concurrent mossy fiber visual and head velocity inputs. Although the climbing fiber inputs to the cerebellum were not directly represented in this model (figure 2), the image velocity signal carried by the mossy fibers to $P$ was used in the model to achieve the same result.

There is good indirect evidence that learning also occurs in the vestibular nucleus. We have adopted the suggestion of Lisberger (1988) that the effectiveness of the head velocity input to some neurons in the vestibular nucleus may be modified by head velocity input in

$$\Delta = \begin{pmatrix} \text{Learning} \\ \text{Rate} \end{pmatrix} \times \begin{pmatrix} \text{Input} \\ \text{Signal} \end{pmatrix} \times \begin{pmatrix} \text{Error} \\ \text{Signal} \end{pmatrix}$$

Cerebellum (P):

$$\dot{A} = \eta_A \times \begin{pmatrix} \text{Head} \\ \text{Velocity} \end{pmatrix} \times \begin{pmatrix} \text{Mossy fiber} \\ \text{Visual signal} \end{pmatrix}$$
$$= \eta_A \times \dot{h} \times -v(g\dot{h} + \dot{e})$$
$$= \eta_A \times \dot{h} \times -v[(g - D)\dot{h} + P]$$
$$\propto \dot{h}^2$$

Vestibular nucleus (N):

$$\dot{D} = \eta_D \times \begin{pmatrix} \text{Head} \\ \text{Velocity} \end{pmatrix} \times \begin{pmatrix} \text{Climbing fiber} \\ \text{Visual signal} \end{pmatrix} - \begin{pmatrix} \text{Purkinje} \\ \text{Signal} \end{pmatrix}$$
$$= \eta_D \times \dot{h} \times [(1 - q)(g\dot{h} + \dot{e}) - qP]$$
$$= \eta_D \times \dot{h} \times [(1 - q)(g - D)\dot{h} + (1 - 2q)P]$$
$$\propto \dot{h}^2$$

where

$$P = \frac{A - bD - (g - D)v}{1 - b + v}\dot{h}$$

Figure 3: **Learning rules for the cerebellum and vestibular nucleus.** The gains $A$ and $D$ change according to the correlation of their input signal and the error signal to the node, as shown for $\Delta$ at the top. The parameter $q$ determines the proportion of learning from Purkinje cell inputs compared to learning from climbing fiber inputs. When $q = 1$, only Purkinje cell inputs drive the adaptation at node $N$; if $q = 0$, learning occurs solely from climbing fiber inputs.

association with Purkinje cells firing. We have also added adaptation from pairing the head velocity input with climbing fiber firing. The relative effectiveness of these two learning mechanisms is controlled by the parameter $q$ (figure 3).

Learning for gain $D$ depends on the interplay between several signals. If the VOR gain is too small, a rightward head turn $P$ (positive value for head velocity) results in too small a leftward eye turn (a negative value for eye velocity). Consequently, the visual scene appears to move to the left (negative image slip). $P$ then fires below its resting level (negative) and its inhibitory influence on $N$ decreases so that $N$ increases its firing rate (figure 4 bottom left). This corrects the VOR gain and increases gain $D$ according to figure 3. Concurrently, the climbing fiber visual signal is above resting firing rate (positive) which also leads to an increase in gain $D$.

Since the signal passing through gain $A$ has an inhibitory influence via $P$ onto $N$, decreasing gain $A$ has the opposite effect on the eye velocity as decreasing gain $D$. Hence, if the VOR is too small we expect gain $A$ to decrease. This is what happens during the early phase of learning (figure 4 top left).

## 4   RESULTS

Finite difference equations of the learning rules were used to calculate changes in gains $A$ and $D$ at the end of each cycle during our simulations. A cycle was defined as one biphasic

Desired gain $g = 1.6$

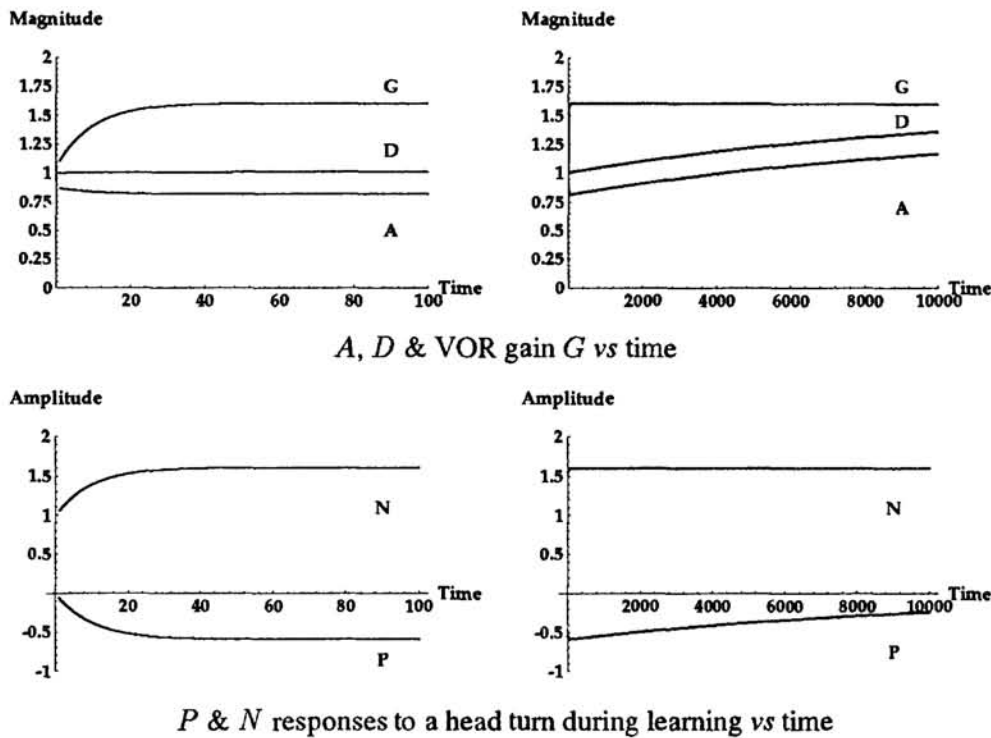

$A, D$ & VOR gain $G$ *vs* time

$P$ & $N$ responses to a head turn during learning *vs* time

Figure 4: **Simulation of change in gain from 1.0 to 1.6.** Top: Short-term (left) and long-term (right) adaptation of the gains $A$, $D$ and $G$. Bottom: Changes on two time scales of $P$ and $N$ responses to a head turn of amplitude 1 during learning. The parameters were $v = 1.0$, $b = .88$, $r = \frac{\eta_A}{\eta_D} = 10.$, and $q = .01$.

head velocity input as shown in figure 2. We assumed that the learning rates were so small that the changes in gains, and hence in the node responses, were negligibly small during each iteration. This allowed the replacement of $A(t)$ and $D(t)$ by their values obtained on the previous iteration for the calculations of $\dot{A}$ and $\dot{D}$. The period of the iteration as well as the amplitude of the head velocity input were chosen so that the integral of the head velocity squared over one iteration equaled 1.

For the simulations shown in figure 4 the gain $G$ of the VOR increased monotonically from 1 to reach the desired value 1.6 within 60 time steps. This rapid adaptation was mainly due to a rapid decrease in $A$, as expected from the local learning rule (figure 3), since the learning rate $\eta_A$ was greater than the learning rate $\eta_D$. Over a longer time period, learning was transferred from $A$ to $D$: $D$ increased from 1 to reach its final value 1.6 while the VOR gain stayed constant. Transfer of learning occurs when $P$ fires in conjunction with a head turn. $P$ can have an elevated firing rate even though the visual error signal is zero (that is, even if the VOR gain $G$ has reached the desired gain $g$) because of the difference between its two other inputs: the head velocity input through $A$ and the eye velocity feedback input through $b$. It is only when these two inputs become equal in amplitude that $P$ firing goes to zero. It can be shown that when learning settles (when $\dot{D}$ and $\dot{A}$ equal zero) $D = g$, $A = bg$, and $P = 0$. With these values for $A$ and $D$, the two other inputs to $P$ are indeed equal in amplitude: one equals $A\dot{h}$, while the other equals $b(-1)D\dot{h}$. During the later part of learning, gain $A$ is driven in the opposite direction (increase) than during the earlier

part (decrease). This comes from a sign reversal of the visual error input to $P$. After the first 60 time steps, the gain has reached the desired gain due to a rapid decrease in $A$, this means that any subsequent increase in $D$, due to transfer of learning as explained above, will cause the gain of the VOR $G$ to become larger than the desired gain $g$, hence the visual error changes sign. In order to compensate for this small error, gain $A$ increases promptly, keeping $G$ very close to the desired gain. This process goes on until $A$ and $D$ reach their equilibrium values stated above.

The short and long-term changes in $P$ and $N$ responses to a velocity step are also shown. As the firing of $P$ decreased with the adaptation of $A$, the firing rate of $N$ increased to the right level.

## 5    OVERSHOOT OF THE VOR GAIN $G$

In this section we show that for some ranges of the learning parameters, the gain $G$ in the model overshoots the desired value $g$. Since an overshoot is not observed in animals (figure 1), this provides constraints on the parameters. The parameter $q$ in the learning rule for the vestibular nucleus (node $N$, gain $D$), determines the proportion of learning from Purkinje cell inputs compared to learning from climbing fiber inputs. When $q = 1$, only Purkinje cell inputs drive the adaptation at node $N$; if $q = 0$, learning at $N$ occurs solely from climbing fiber inputs. These two inputs have quite different effects on learning as shown in figure 5. Asymptotically, $P$ goes to 0, and $D$ goes to $g$ if $q = 1$; and $P$ can only differ from 0 if $q = 0$. The gain has an overshoot for any value of $q$ different than 0, as shown in figure 6. Nevertheless, its amplitude is only significant for a limited extent in the parameter space of $q$ and $r$ (graph of figure 6). The overshoot is reduced with a smaller $q$ and a larger $r$. One possibility is that $q$ is chosen close to 0 and $r \gg 1$, that is $\eta_A \gg \eta_D$. These conditions were used to choose parameter values in the simulations (figure 4).

## 6    DISCUSSION AND CONCLUSION

The VOR model analyzed here is a static model without time delays and multiple time scales. We are currently studying how these factors affect the time course of learning in a dynamical model of the VOR and smooth pursuit.

In our model, learning occurs in the dark if $P \neq 0$, which has not been observed in animals. One way to avoid learning in the dark when $P$ is firing would be to gate the learning by a visual input, such as that provided by climbing fibers.

The responses of vestibular afferents to head motion can be classified into two categories: phase-tonic and tonic. In this model, only the tonic afferents were represented. Both afferent types encode head velocity, while the phasic-tonic responds to head acceleration as well. The steady state VOR gain can also be changed by altering the relative proportions of phasic and tonic afferents to the Purkinje cells (Lisberger and Sejnowski, 1992). We are currently investigating learning rules for which this occurs.

The model predicts that adaptation in the cerebellum is faster than in the vestibular nucleus, and that learning in the vestibular nucleus is mostly driven by the climbing fiber error signals.

The model shows how the dynamics of the whole system can lead to long-term adaptation

Desired gain $g = 1.6$

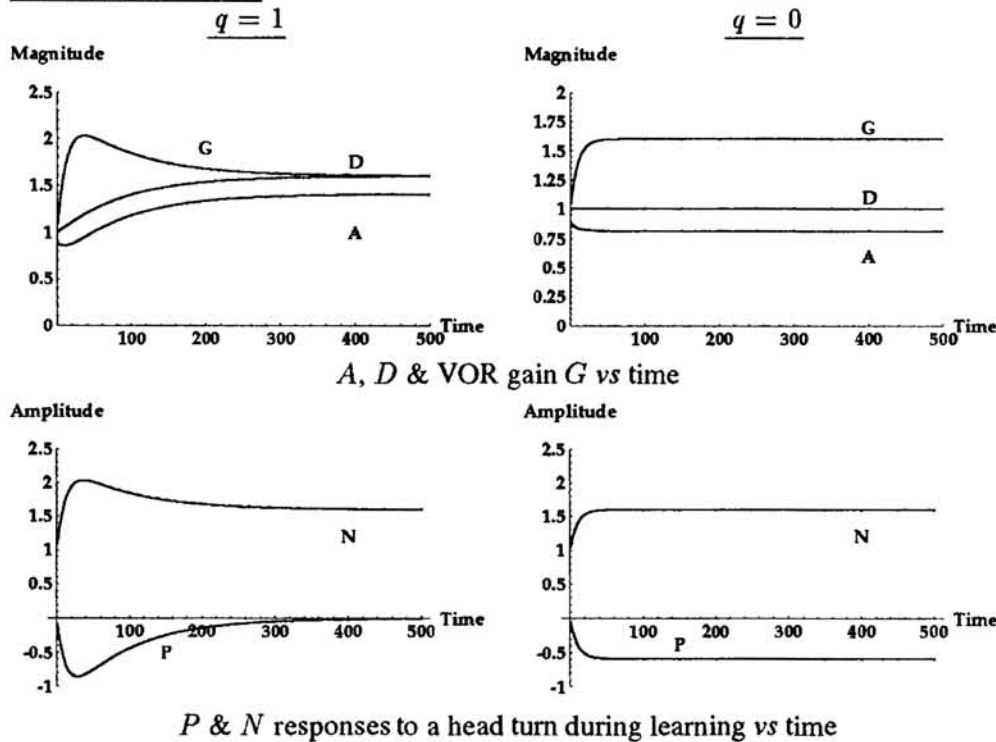

$$\epsilon = \frac{(1-b+v)}{(1-b)}(D-g)\frac{q}{(2q-1)-rv}$$

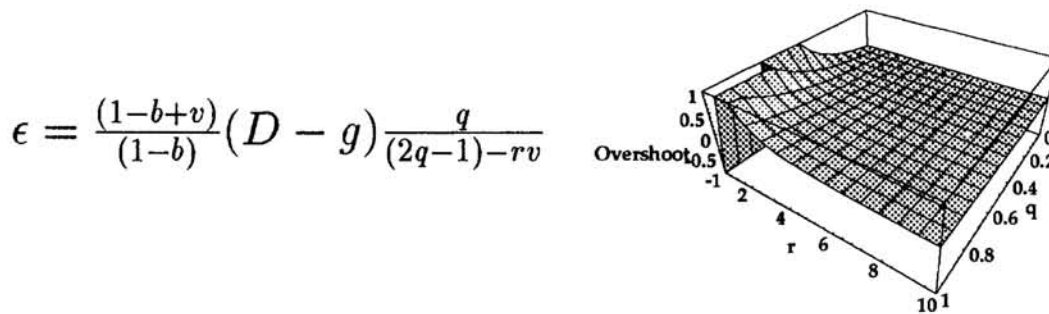

Figure 5: **Effect of $q$ on learning curves for gain increase.** Left: $q = 1$ leads to an overshoot in the VOR gain $G$ above the desired gain. $D$ increases up to the desired gain, $P$ starts from 0 and asymptotically goes back to 0; both indicate that learning is totally transferred from $P$ to $N$. Right: With $q = 0$, there is no overshoot in the VOR gain, but since $A$ decreases to a constant value and $D$ only increases very slightly, learning is not transfered. Consequently, $P$ firing rate stays constant after an initial drop.

Figure 6: **Overshoot $\epsilon$ of the VOR gain $G$ as a function of $q$ and $r$.** The parameter $q$ is the proportion of learning to node $N$ (vestibular nucleus), coming from the $P$ node (cerebellum) compared to learning from climbing fibers. The parameter $r$ is the ratio of the learning rates $\eta_A$ and $\eta_D$. No overshoot is seen in animals, which restricts the parameters space of $q$ and $r$ for the model to be valid. Note that the overshoot diverges for some parameter values.

which differs from what may be expected from the local learning rules at the synapses because of differences in time scales and shifts of activity in the system during learning. This may reconcile apparently contradictory evidence between local learning rules observed *in vitro* (Ito,1970) and the long-term adaptation seen *in vivo* in animals (Miles and Lisberger,1981).

## Acknowledgments

O.C. was supported by NSERC during this research.

## References

Albus, J. S. (1971). A theory of cerebellar function. *Math. Biosci.*, 10:25–61.

Arnold, D. B. and Robinson, D. A. (1992). A neural network model of the vestibulo-ocular reflex using a local synaptic learning rule. *Phil. Trans. R. Soc. Lond. B*, 337:327–330.

Fujita, M. (1982). Simulations of adaptive modification of the vestibulo-ocular reflex with an adaptive filter model of the cerebellum. *Biological Cybernetics*, 45:207–214.

Galiana, H. L. (1986). A new approach to understanding adaptive visual-vestibular interactions in the central nervous system. *Journal of Neurophysiology*, 55:349–374.

Ito, M. (1970). Neurophysiological aspects of the cerebellar motor control system. *Int.J.Neurol.*, 7:162–176.

Ito, M., Sakurai, M., and Tongroach, P. (1982). Climbing fibre induced depression of both mossy fibre responsiveness and glutamate sensitivity of cerebellar purkinje cells. *J. Physiol. Lond.*, 324:113–134.

Kawato, M. and Gomi, H. (1992). The cerebellum and VOR/OKR learning models. *Trends in Neuroscience*, 15:445–453.

Lisberger, S. G. (1988). The neural basis for learning of simple motor skills. *Science*, 242:728–735.

Lisberger, S. G. (1992). Neural basis for motor learning in the vestibulo-ocular reflex of primates:IV. The sites of learning. In preparation.

Lisberger, S. G. and Sejnowski, T. J. (1992). Computational analysis suggests a new hypothesis for motor learning in the vestibulo-ocular reflex. Technical Report 9201, INC, Univ. of California, San Diego.

Marr, D. (1969). A theory of cerebellar cortex. *J. Physiol.*, 202:437–470.

Melvill Jones, G. M. (1991). *The Vestibular Contribution*, volume 8 of *Vision and Visual Dysfunction*, chapter 2, pages 293–303. CRC Press, Inc., Boston. General Editor: J. R. Cronly-Dillon.

Miles, F. A. and Eighmy, B. B. (1980). Long-term adaptive changes in primate vestibulo-ocular reflex.I. Behavioural observations. *Journal of Neurophysiology*, 43:1406–1425.

Miles, F. A. and Lisberger, S. G. (1981). Plasticity in the vestibulo-ocular reflex: A new hypothesis. *Ann. Rev. Neurosci.*, 4:273–299.

Quinn, K. J., Baker, J., and Peterson, B. (1992). Simulation of cerebellar-vestibular interactions during VOR adaptation. In *Program 22nd Annual Meeting*. Society for Neuroscience.

Quinn, K. J., Schmajuk, N., Jain, A., Baker, J. F., and Peterson, B. W. (1992). Vestibuloocular reflex arc analysis using an experimentally constrained network. *Biological Cybernetics*, 67:113–122.

Viola, P. A., Lisberger, S. G., and Sejnowski, T. J. (1992). Recurrent eye tracking network using a distributed representation of image motion. In Moody, J. E., Hansen, S. J., and Lippman, R. P., editors, *Advances in Neural Information Processing Systems 4*, San Mateo. IEEE, Morgan Kaufmann Publishers.